# Phone Recognition with the Mean-Covariance Restricted Boltzmann Machine

**George E. Dahl, Marc'Aurelio Ranzato, Abdel-rahman Mohamed, and Geoffrey Hinton**
Department of Computer Science
University of Toronto
{gdahl, ranzato, asamir, hinton}@cs.toronto.edu

## Abstract

Straightforward application of Deep Belief Nets (DBNs) to acoustic modeling produces a rich distributed representation of speech data that is useful for recognition and yields impressive results on the speaker-independent TIMIT phone recognition task. However, the first-layer Gaussian-Bernoulli Restricted Boltzmann Machine (GRBM) has an important limitation, shared with mixtures of diagonal-covariance Gaussians: GRBMs treat different components of the acoustic input vector as conditionally independent given the hidden state. The mean-covariance restricted Boltzmann machine (mcRBM), first introduced for modeling natural images, is a much more representationally efficient and powerful way of modeling the covariance structure of speech data. Every configuration of the precision units of the mcRBM specifies a different precision matrix for the conditional distribution over the acoustic space. In this work, we use the mcRBM to learn features of speech data that serve as input into a standard DBN. The mcRBM features combined with DBNs allow us to achieve a phone error rate of 20.5%, which is superior to all published results on speaker-independent TIMIT to date.

## 1  Introduction

Acoustic modeling is a fundamental problem in automatic continuous speech recognition. Most state of the art speech recognition systems perform acoustic modeling using the following approach [1]. The acoustic signal is represented as a sequence of feature vectors; these feature vectors typically hold a log spectral estimate on a perceptually warped frequency scale and are augmented with the first and second (at least) temporal derivatives of this spectral information, computed using smoothed differences of neighboring frames. Hidden Markov models (HMMs), with Gaussian mixture models (GMMs) for the emission distributions, are used to model the probability of the acoustic vector sequence given the (tri)phone sequence in the utterance to be recognized.[1] Typically, all of the individual Gaussians in the mixtures are restricted to have diagonal covariance matrices and a large hidden Markov model is constructed from sub-HMMs for each triphone to help deal with the effects of context-dependent variations. However, to mitigate the obvious data-sparsity and efficiency problems context dependence creates, modern systems perform sophisticated parameter tying by clustering the HMM states using carefully constructed decision trees to make state tying choices.

Although systems of this sort have yielded many useful results, diagonal covariance CDHMM models have several potential weaknesses as models of speech data. On the face of things at least, feature vectors for overlapping frames are treated as independent and feature vectors must be augmented with derivative information in order to enable successful modeling with mixtures of diagonal-covariance Gaussians (see [2, 3] for a more in-depth discussion of the exact consequences of the delta features). However, perhaps even more disturbing than the frame-independence assumption are the compromises required to deal with two competing pressures in Gaussian mixture model

training: the need for expressive models capable of representing the variability present in real speech data and the need to combat the resulting data sparsity and statistical efficiency issues. These pressures of course exist for other models as well, but the tendency of GMMs to partition the input space into regions where only one component of the mixture dominates is a weakness that inhibits efficient use of a very large number of tunable parameters. The common decision to use diagonal covariance Gaussians for the mixture components is an example of such a compromise of expressiveness that suggests that it might be worthwhile to explore models in which each parameter is constrained by a large fraction of the training data. By contrast, models that use the simultaneous activation of a large number of hidden features to generate an observed input can use many more of their parameters to model each training example and hence have many more training examples to constrain each parameter. As a result, models that use non-linear distributed representations are harder to fit to data, but they have much more representational power for the same number of parameters.

The diagonal covariance approximation typically employed for GMM-based acoustic models is symptomatic of, but distinct from, the general representational inefficiencies that tend to crop up in mixture models with massive numbers of highly specialized, distinctly parameterized mixture components. Restricting mixture components to have diagonal covariance matrices introduces a conditional independence assumption between dimensions within a single frame. The delta-feature augmentation mitigates the severity of the approximation and thus makes outperforming diagonal covariance Gaussian mixture models difficult. However, a variety of precision matrix modeling techniques have emerged in the speech recognition literature. For example, [4] describes a basis superposition framework that includes many of these techniques.

Although the recent work in [5] on using deep belief nets (DBNs) for phone recognition begins to attack the representational efficiency issues of GMMs, Gaussian-Bernoulli Restricted Boltzmann Machines (GRBMs) are used to deal with the real-valued input representation (in this case, mel-frequency cepstral coefficients). GRBMs model different dimensions of their input as conditionally independent given the hidden unit activations, a weakness akin to restricting Gaussians in a GMM to have diagonal covariance. This conditional independence assumption is inappropriate for speech data encoded as a sequence of overlapping frames of spectral information, especially when many frames are concatenated to form the input vector. Such data can exhibit local smoothness in both frequency and time punctuated by bursts of energy that violate these local smoothness properties. Performing a standard augmentation of the input with temporal derivative information, as [5] did, will of course make it easier for GRBMs to deal with such data, but ideally one would use a model capable of *succinctly* modeling these effects on its own.

Inspired by recent successes in modeling natural images, the primary contribution of this work is to bring the mean-covariance restricted Boltzmann machine (mcRBM) of [6] to bear on the problem of extracting useful features for phone recognition and to incorporate these features into a deep architecture similar to one described in [5]. We demonstrate the efficacy of our approach by reporting results on the speaker-independent TIMIT phone recognition task. TIMIT, as argued in [7], is an ideal dataset for testing new ideas in speech recognition before trying to scale them up to large vocabulary tasks because it is phonetically rich, has well-labeled transcriptions, and is small enough not to pose substantial computational challenges at test time. Our best system achieves a phone error rate on the TIMIT corpus of 20.5%, which is superior to all published results on speaker-independent TIMIT to date. We obtain these results without augmenting the input with temporal difference features since a sensible model of speech data should be able to learn to extract its own useful features that make explicit inclusion of difference features unnecessary.

## 2  Using Deep Belief Nets for Phone Recognition

Following the approach of [5], we use deep belief networks (DBNs), trained via the unsupervised pretraining algorithm described in [8], combined with supervised fine-tuning using backpropagation, to model the posterior distribution over HMM states given a local window of the acoustic input. We construct training cases for the DBN by taking $n$ adjacent frames of acoustic input and pairing them with the identity of the HMM state for the central frame. We obtain the labels from a forced alignment with a CDHMM baseline. During the supervised phase of learning, we optimize the cross-entropy loss for the individual HMM-state predictions, as a more convenient proxy for the number of mistakes (insertions, deletions, substitutions) in the phone sequence our system produces, which

is what we are actually interested in. In order to compare with the results [5], at test time, we use the posterior probability distribution over HMM states that the DBN produces in place of GMM likelihoods in an otherwise standard Viterbi decoder. Since the HMM defines a prior over states, it is better to divide the posterior probabilities of the DBN by the frequencies of the 183 labels in the training data [9], but in our experiments this did not noticeably change the results.

## 3  The Mean-Covariance Restricted Boltzmann Machine

The previous work of [5] used a GRBM for the initial DBN layer. The GRBM associates each configuration of the visible units, $\mathbf{v}$, and hidden units, $\mathbf{h}$, with a probability density according to

$$P(\mathbf{v}, \mathbf{h}) \propto e^{-E(\mathbf{v},\mathbf{h})}, \tag{1}$$

where $E(\mathbf{v}, \mathbf{h})$ is given by

$$E(\mathbf{v}, \mathbf{h}) = \frac{1}{2}(\mathbf{v} - \mathbf{b})^{\mathrm{T}}(\mathbf{v} - \mathbf{b}) - \mathbf{c}^{\mathrm{T}}\mathbf{h} - \mathbf{v}^{\mathrm{T}}\mathbf{W}\mathbf{h}, \tag{2}$$

and where $\mathbf{W}$ is the matrix of visible/hidden connection weights, $\mathbf{b}$ is a visible unit bias, and $\mathbf{c}$ is a hidden unit bias. Equation 2 implicitly assumes that the visible units have a diagonal covariance Gaussian noise model with a variance of 1 on each dimension.

Another option for learning to extract binary features from real-valued data that has enjoyed success in vision applications is the mean-covariance RBM (mcRBM), first introduced in [10] and [6]. The mcRBM has two groups of hidden units: mean units and precision units. Without the precision units, the mcRBM would be identical to a GRBM. With only the precision units, we have what we will call the "cRBM", following the terminology in [6]. The precision units are designed to enforce smoothness constraints in the data, but when one of these constraints is seriously violated, it is removed by turning off the precision unit. The set of active precision units therefore specifies a sample-specific covariance matrix. In order for a visible vector to be assigned high probability by the precision units, it must only fail to satisfy a small number of the precision unit constraints, although each of these constraints could be egregiously violated.

The cRBM can be viewed as a particular type of factored third order Boltzmann machine. In other words, the RBM energy function is modified to have multiplicative interactions between triples of two visible units, $v_i$ and $v_j$, and one hidden unit $h_k$. Unrestricted 3-way connectivity causes a cubic growth in the number of parameters that is unacceptable if we wish to scale this sort of model to high dimensional data. Factoring the weights into a sum of 3-way outer products can reduce the growth rate of the number of parameters in the model to one that is comparable to a normal RBM. After factoring, we may write the cRBM energy function[2] (with visible biases omitted) as:

$$E(\mathbf{v}, \mathbf{h}) = -\mathbf{d}^{\mathrm{T}}\mathbf{h} - (\mathbf{v}^{\mathrm{T}}\mathbf{R})^2\mathbf{P}\mathbf{h}, \tag{3}$$

where $\mathbf{R}$ is the visible-factor weight matrix, $\mathbf{d}$ denotes the hidden unit bias vector, and $\mathbf{P}$ is the factor-hidden, or "pooling" matrix. The squaring in equation 3 (and in other equations with this term) is performed elementwise. We force $\mathbf{P}$ to only have non-positive entries. We must constrain $\mathbf{P}$ in this way to avoid a model that assigns larger and larger probabilities (more negative energies) to larger and larger inputs.

The hidden units of the cRBM are still (just as in GRBMs) conditionally independent given the states of the visible units, so inference remains simple. However, the visible units are coupled in a Markov Random Field determined by the settings of the hidden units. The interaction weight between two arbitrary visible units $v_i$ and $v_j$, which we shall denote $\tilde{w}_{i,j}$, depends on the states of all the hidden units according to:

$$\tilde{w}_{i,j} = \sum_k \sum_f h_k r_{if} r_{jf} p_{kf}.$$

The conditional distribution of the hidden units (derived from 3) given the visible unit states $\mathbf{v}$ is:

$$P(\mathbf{h}|\mathbf{v}) = \sigma\left(\mathbf{d} + \left((\mathbf{v}^{\mathrm{T}}\mathbf{R})^2\mathbf{P}\right)^{\mathrm{T}}\right),$$

where $\sigma$ denotes the elementwise logistic sigmoid, $\sigma(x) = (1+e^{-x})^{-1}$. The conditional distribution of the visible units given the hidden unit states for the cRBM is given by:

$$P(\mathbf{v}|\mathbf{h}) \sim \mathcal{N}\left(\mathbf{0}, \left[\mathbf{R}\left(\text{diag}(-\mathbf{P}^{\mathrm{T}}\mathbf{h})\right)\mathbf{R}^{\mathrm{T}}\right]^{-1}\right). \tag{4}$$

The cRBM always assigns highest probability to the all zero visible vector. In order to allow the model to shift the mean, we add an additional set of binary hidden units whose vector of states we shall denote $\mathbf{m}$. The product of the distributions defined by the cRBM and the GRBM forms the mcRBM. If $E_C(\mathbf{v}, \mathbf{h})$ denotes the cRBM energy function (equation 3) and $E_M(\mathbf{v}, \mathbf{m})$ denotes the GRBM energy function (equation 2), then the mcRBM energy function is:

$$E_{MC}(\mathbf{v}, \mathbf{h}, \mathbf{m}) = E_C(\mathbf{v}, \mathbf{h}) + E_M(\mathbf{v}, \mathbf{m}). \tag{5}$$

The gradient of the $E_M$ term moves the minimum of $E_{MC}$ away from the zero vector, but how far it moves depends on the curvature of the precision matrix defined by $E_C$. The resulting conditional distribution over the visible units, given the two sets of hidden units is:

$$P(\mathbf{v}|\mathbf{h}, \mathbf{m}) \propto \mathcal{N}\left(\Sigma\mathbf{W}\mathbf{m}, \Sigma\right),$$

where

$$\Sigma = \left(\mathbf{R}\left(\text{diag}(-\mathbf{P}^{\mathrm{T}}\mathbf{h})\right)\mathbf{R}^{\mathrm{T}}\right)^{-1}.$$

Thus the mcRBM can produce conditional distributions over the visible units, given the hidden units, that have non-zero means, unlike the cRBM.

Just like other RBMs, the mcRBM can be trained using the following update rule, for some generic model parameter $\theta$:

$$\Delta\theta \propto \langle -\frac{\partial E}{\partial \theta}\rangle_{data} + \langle \frac{\partial E}{\partial \theta}\rangle_{reconstruction}.$$

However, since the matrix inversion required to sample from $P(\mathbf{v}|\mathbf{h}, \mathbf{m})$ can be expensive, we integrate out the hidden units and use Hybrid Monte Carlo (HMC) [11] on the mcRBM free energy to obtain the reconstructions.

It is important to emphasize that the mcRBM model of covariance structure is much more powerful than merely learning a covariance matrix in a GRBM. Learning the covariance matrix for a GRBM is equivalent to learning a single global linear transformation of the data, whereas the precision units of an mcRBM are capable of specifying exponentially many different covariance matrices and explaining different visible vectors with different distributions over these matrices.

### 3.1 Practical details

In order to facilitate stable training, we make the precision unit term in the energy function insensitive to the scale of the input data by normalizing by the length of $\mathbf{v}$. This makes the conditional $P(\mathbf{v}|\mathbf{h})$ clearly non-Gaussian. We constrain the columns of $\mathbf{P}$ to have unit L1 norm and to be sparse. We enforce one-dimensional locality and sparsity in $\mathbf{P}$ by setting entries beyond a distance of one from the main diagonal to zero after every update. Additionally, we constrain the columns of $\mathbf{R}$ to all have equal L2 norms and learn a single global scaling factor shared across all the factors. The non-positivity constraint on the entries of $\mathbf{P}$ is maintained by zeroing out, after each update, any entries that become positive.

## 4 Deep Belief Nets

Learning is difficult in densely connected, directed belief nets that have many hidden layers because it is difficult to infer the posterior distribution over the hidden variables, when given a data vector, due to the phenomenon of explaining away. Markov chain Monte Carlo methods [12] can be used to sample from the posterior, but they are typically very time-consuming. In [8] complementary priors were used to eliminate the explaining away effects, producing a training procedure which is equivalent to training a stack of restricted Boltzmann machines.

The stacking procedure works as follows. Once an RBM has been trained on data, we can infer the hidden unit activation probabilities given a data vector and re-represent the data vector as the vector of corresponding hidden activations. Since the RBM has been trained to reconstruct the data

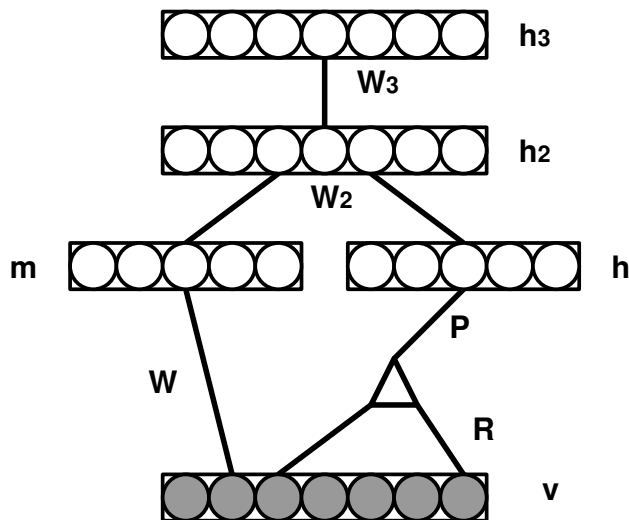

Figure 1: An mcRBM with two RBMs stacked on top

well, the hidden unit activations will retain much of the information present in the data and pick up (possibly higher-order) correlations between different data dimensions that exist in the training set. Once we have used one RBM as a feature extractor we can, if desired, train an additional RBM that treats the hidden activations of the first RBM as data to model. After training a sequence of RBMs, we can compose them to form a generative model whose top two layers are the final RBM in the stack and whose lower layers all have downward-directed connections that implement the $p(\mathbf{h}_{k-1}|\mathbf{h}_k)$ learned by the $k^{th}$ RBM, where $\mathbf{h}_0 = \mathbf{v}$.

The weights obtained by the greedy layer-by-layer training procedure described for stacking RBMs, above, can be used to initialize the weights of a deep feed-forward neural network. Once we add an output layer to the pre-trained neural network, we can discriminatively fine-tune the weights of this neural net using any variant of backpropagation [13] we wish. Although options for fine-tuning exist other than backpropagation, such as the up-down algorithm used in [8], we restrict ourselves to backpropagation (updating the weights every 128 training cases) in this work for simplicity and because it is sufficient for obtaining excellent results.

Figure 1 is a diagram of two RBMs stacked on top of an mcRBM. Note that the RBM immediately above the mcRBM uses both the mean unit activities and the precision unit activities together as visible data. Later, during backpropagation, after we have added the softmax output unit, we do not backpropagate through the mcRBM weights, so the mcRBM is a purely unsupervised feature extractor.

## 5 Experimental Setup

### 5.1 The TIMIT Dataset

We used the TIMIT corpus[3] for all of our phone recognition experiments. We used the 462 speaker training set and removed all SA records (i.e., identical sentences for all speakers in the database), since they could potentially bias our results. A development set of 50 speakers was used for hand-tuning hyperparameters and automated decoder tuning. As is standard practice, results are reported using the 24-speaker core test set. We produced the training labels with a forced alignment of an HMM baseline. Since there are three HMM states per phone and 61 phones, all DBN architectures had a 183-way softmax output unit. Once the training labels have been created, the HMM baseline

is no longer needed; we do not combine or average our results with any HMM+GMM system. After decoding, starting and ending silences were removed and the 61 phone classes were mapped to a set of 39 classes as in [14] for scoring. We removed starting and ending silences before scoring in order to be as similar to [5] as possible. However, to produce a more informative comparison between our results and results in the literature that do not remove starting and ending silences, we also present the phone error rate of our best model using the more common scoring strategy. During decoding, we used a simple bigram language model over phones. Our results would certainly improve with a trigram language model. In order to be able to make useful comparisons between different DBN architectures (and achieve the best results), we optimized the Viterbi decoder parameters (the word insertion probability and the language model scale factor) on the development set and then used the best performing setting to compute the phone error rate (PER) for the core test set.

## 5.2 Preprocessing

Since we have completely abandoned Gaussian mixture model emission distributions, we are no longer forced to use temporal derivative features. For all experiments the acoustic signal was analyzed using a 25-ms Hamming window with 10-ms between the left edges of successive frames. We use the output from a mel scale filterbank, extracting 39 filterbank output log magnitudes and one log energy per frame. Once groups of 15 frames have been concatenated, we perform PCA whitening and preserve the 384 most important principal components. Since we perform PCA whitening anyway, the discrete cosine transform used to compute mel frequency cepstral coefficients (MFCCs) from the filterbank output is not useful. Determining the number of frames of acoustic context to give to the DBN is an important preprocessing decision; preliminary experiments revealed that moving to 15 frames of acoustic data, from the 11 used in [5], could provide improvements in PER when training a DBN on features from a mcRBM. It is possible that even larger acoustic contexts might be beneficial as well. Also, since the mcRBM is trained as a generative model, doubling the input dimensionality by using a 5-ms advance per frame is unlikely to cause serious overfitting and might well improve performance.

## 5.3 Computational Setup

Training DBNs of the sizes used in this paper can be computationally expensive. We accelerated training by exploiting graphics processors, in particular GPUs in a NVIDIA Tesla S1070 system, using the wonderful library described in [15]. The wall time per epoch varied with the architecture. An epoch of training of an mcRBM that had 1536 hidden units (1024 precision units and 512 mean units) took 20 minutes. When each DBN layer had 2048 hidden units, each epoch of pre-training for the first DBN layer took about three minutes and each epoch of pretraining for the fifth layer took seven to eight minutes, since we propagated through each earlier layer. Each epoch of fine-tuning for such a five-DBN-layer architecture took 12 minutes. We used 100 epochs to train the mcRBM, 50 epochs to train each RBM in the stack and 14 epochs of discriminative fine-tuning of the whole network for a total of nearly 60 hours, about 34 of which were spent training the mcRBM.

# 6 Experiments

Since one goal of this work is to improve performance on TIMIT by using deep learning architectures, we explored varying the number of DBN layers in our architecture. In agreement with [5], we found that in order to obtain the best results with DBNs on TIMIT, multiple layers were essential.

Figure 2 plots phone error rate on both the development set and the core test set against the number of hidden layers in a mcRBM-DBN (we don't count the mcRBM as a hidden layer since we do not backpropagate through it). The particular mcRBM-DBN shown had 1536 hidden units in each DBN hidden layer, 1024 precision units in the mcRBM, and 512 mean units in the mcRBM. As the number of DBN hidden layers increased, error on the development and test sets decreased and eventually leveled off. The improvements that deeper models can provide over shallower models were evident from results reported in [5]; the results for the mcRBM-DBN in this work are even more dramatic. In fact, an mcRBM-DBN with 8 hidden layers is what exhibits the best development set error, 20.17%, in these experiments. The same model gets 21.7% on the core test set (20.5% if starting and ending silences are included in scoring). Furthermore, at least 5 DBN hidden layers seem to be necessary

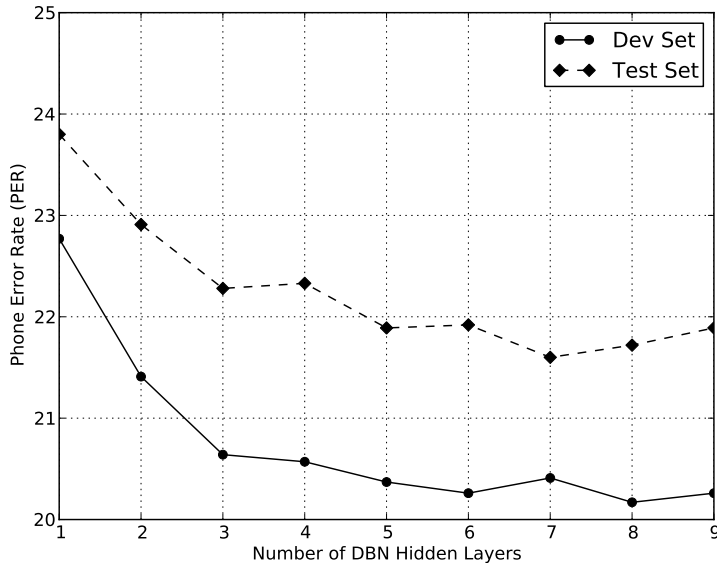

Figure 2: Effect of increasing model depth

Table 1: *The effect of DBN layer size on Phone Error Rate for 5 layer mcRBM-DBN models*

| Model | devset | testset |
|-------|--------|---------|
| 512 units | 21.4% | 22.8% |
| 1024 units | 20.9% | 22.3% |
| 1536 units | 20.4% | 21.9% |
| 2048 units | 20.4% | 21.8% |

to break a test set PER of 22%. Models of this depth (note also that an mcRBM-DBN with 8 DBN hidden layers is really a 9 layer model) have rarely been employed in the deep learning literature (cf. [8, 16], for example).

Table 1 demonstrates that once the hidden layers are sufficiently large, continuing to increase the size of the hidden layers did not seem to provide additional improvements. In general, we did not find our results to be very sensitive to the exact number of hidden units in each layer, as long the hidden layers were relatively large.

To isolate the advantage of using an mcRBM instead of a GRBM, we need a clear comparison that is not confounded by the differences in preprocessing between our work and [5]. Table 2 provides such a comparison and confirms that the mcRBM feature extraction causes a noticeable improvement in PER. The architectures in table 2 use 1536-hidden-unit DBN layers.

Table 3 compares previously published results on the speaker-independent TIMIT phone recognition task to the best mcRBM-DBN architecture we investigated. Results marked with a * remove starting

Table 2: *mcRBM-DBN vs GRBM-DBN Phone Error Rate*

| Model | devset PER | testset PER |
|-------|------------|-------------|
| 5 layer GRBM-DBN | 22.3% | 23.7% |
| mcRBM + 4 layer DBN | 20.6% | 22.3% |

Table 3: *Reported (speaker independent) results on TIMIT core test set*

| Method | PER |
|---|---|
| Stochastic Segmental Models [17] | 36% |
| Conditional Random Field [18] | 34.8% |
| Large-Margin GMM [19] | 33% |
| CD-HMM [20] | 27.3% |
| Augmented conditional Random Fields [20] | 26.6% |
| Recurrent Neural Nets [21] | 26.1% |
| Bayesian Triphone HMM [22] | 25.6% |
| Monophone HTMs [23] | 24.8% |
| Heterogeneous Classifiers [24] | 24.4% |
| Deep Belief Networks(DBNs) [5] | 23.0*% |
| Triphone HMMs discriminatively trained w/ BMMI [7] | 22.7% |
| Deep Belief Networks with mcRBM feature extraction (this work) | **21.7*%** |
| Deep Belief Networks with mcRBM feature extraction (this work) | **20.5%** |

and ending silences at test time before scoring. One should note that the work of [7] used triphone HMMs and a trigram language model whereas in this work we used only a bigram language model and monophone HMMs, so table 3 probably underestimates the error reduction our system provides over the best published GMM-based approach.

## 7  Conclusions and Future Work

We have presented a new deep architecture for phone recognition that combines a mcRBM feature extraction module with a standard DBN. Our approach attacks both the representational inefficiency issues of GMMs and an important limitation of previous work applying DBNs to phone recognition. The incorporation of features extracted by a mcRBM into an approach similar to that of [5] produces results on speaker-independent TIMIT superior to those that have been reported to date. However, DBN-based acoustic modeling approaches are still in their infancy and many important research questions remain. During the fine-tuning, one could imagine backpropagating through the decoder itself and optimizing an objective function more closely related to the phone error rate. Since the pretraining procedure can make use of large quantities of completely unlabeled data, leveraging untranscribed speech data on a large scale might allow our approach to be even more robust to inter-speaker acoustic variations and would certainly be an interesting avenue of future work.

## Footnotes

[1]We will refer to HMMs with GMM emission distributions as CDHMMs for continuous-density HMMs.

[2]In order to normalize the distribution implied by this energy function, we must restrict the visible units to a region of the input space that has finite extent. However, once we add the mean RBM this normalization issue vanishes.

[3]http://www.ldc.upenn.edu/Catalog/CatalogEntry.jsp?catalogId=LDC93S1.

## References

[1] S. Young, "Statistical modeling in continuous speech recognition (CSR)," in *UAI '01: Proceedings of the 17th Conference in Uncertainty in Artificial Intelligence*, San Francisco, CA, USA, 2001, pp. 562–571, Morgan Kaufmann Publishers Inc.

[2] C. K. I. Williams, "How to pretend that correlated variables are independent by using difference observations," *Neural Comput.*, vol. 17, no. 1, pp. 1–6, 2005.

[3] J.S. Bridle, "Towards better understanding of the model implied by the use of dynamic features in HMMs," in *Proceedings of the International Conference on Spoken Language Processing*, 2004, pp. 725–728.

[4] K. C. Sim and M. J. F. Gales, "Minimum phone error training of precision matrix models," *IEEE Transactions on Audio, Speech & Language Processing*, vol. 14, no. 3, pp. 882–889, 2006.

[5] A. Mohamed, G. E. Dahl, and G. E. Hinton, "Deep belief networks for phone recognition," in *NIPS Workshop on Deep Learning for Speech Recognition and Related Applications*, 2009.

[6] M. Ranzato and G. Hinton, "Modeling pixel means and covariances using factorized third-order boltzmann machines," in *Proc. of Computer Vision and Pattern Recognition Conference (CVPR 2010)*, 2010.

[7] T. N. Sainath, B. Ramabhadran, and M. Picheny, "An exploration of large vocabulary tools for small vocabulary phonetic recognition," in *IEEE Automatic Speech Recognition and Understanding Workshop*, 2009.

[8] G. E. Hinton, S. Osindero, and Y. Teh, "A fast learning algorithm for deep belief nets," *Neural Computation*, vol. 18, pp. 1527–1554, 2006.

[9] N. Morgan and H. Bourlard, "Continuous speech recognition," *Signal Processing Magazine, IEEE*, vol. 12, no. 3, pp. 24 –42, may 1995.

[10] M. Ranzato, A. Krizhevsky, and G. Hinton, "Factored 3-way restricted Boltzmann machines for modeling natural images," in *Proceedings of the International Conference on Artificial Intelligence and Statistics*, 2010, vol. 13.

[11] R. M. Neal, *Bayesian Learning for Neural Networks*, Springer-Verlag New York, Inc., Secaucus, NJ, USA, 1996.

[12] R. M. Neal, "Connectionist learning of belief networks," *Artificial Intelligence*, vol. 56, no. 1, pp. 71–113, 1992.

[13] D. E. Rumelhart, G. E. Hinton, and R. J. Williams, "Learning representations by back-propagating errors," *Nature*, vol. 323, no. 6088, pp. 533–536, 1986.

[14] K. F. Lee and H. W. Hon, "Speaker-independent phone recognition using hidden markov models," *IEEE Transactions on Audio, Speech & Language Processing*, vol. 37, no. 11, pp. 1641–1648, 1989.

[15] V. Mnih, "Cudamat: a CUDA-based matrix class for python," Tech. Rep. UTML TR 2009-004, Department of Computer Science, University of Toronto, November 2009.

[16] V. Nair and G. E. Hinton, "3-d object recognition with deep belief nets," in *Advances in Neural Information Processing Systems 22*, Y. Bengio, D. Schuurmans, J. Lafferty, C. K. I. Williams, and A. Culotta, Eds., 2009, pp. 1339–1347.

[17] V. V. Digalakis, M. Ostendorf, and J. R. Rohlicek, "Fast algorithms for phone classification and recognition using segment-based models," *IEEE Transactions on Signal Processing*, vol. 40, pp. 2885–2896, 1992.

[18] J. Morris and E. Fosler-Lussier, "Combining phonetic attributes using conditional random fields," in *Proc. Interspeech*, 2006, pp. 597–600.

[19] F. Sha and L. Saul, "Large margin gaussian mixture modeling for phonetic classification and recognition," in *Proc. ICASSP*, 2006, pp. 265–268.

[20] Y. Hifny and S. Renals, "Speech recognition using augmented conditional random fields," *IEEE Transactions on Audio, Speech & Language Processing*, vol. 17, no. 2, pp. 354–365, 2009.

[21] A. Robinson, "An application of recurrent nets to phone probability estimation," *IEEE Transactions on Neural Networks*, vol. 5, no. 2, pp. 298–305, 1994.

[22] J. Ming and F. J. Smith, "Improved phone recognition using bayesian triphone models," in *Proc. ICASSP*, 1998, pp. 409–412.

[23] L. Deng and D. Yu, "Use of differential cepstra as acoustic features in hidden trajectory modelling for phonetic recognition," in *Proc. ICASSP*, 2007, pp. 445–448.

[24] A. Halberstadt and J. Glass, "Heterogeneous measurements and multiple classifiers for speech recognition," in *Proc. ICSLP*, 1998.

